# Search for Information Bearing Components in Speech

**Howard Hua Yang and Hynek Hermansky**
Department of Electrical and Computer Engineering
Oregon Graduate Institute of Science and Technology
20000 NW, Walker Rd., Beaverton, OR97006, USA
{hyang,hynek}@ece.ogi.edu, FAX:503 7481406

## Abstract

In this paper, we use mutual information to characterize the distributions of phonetic and speaker/channel information in a time-frequency space. The mutual information (MI) between the phonetic label and one feature, and the joint mutual information (JMI) between the phonetic label and two or three features are estimated. The Miller's bias formulas for entropy and mutual information estimates are extended to include higher order terms. The MI and the JMI for speaker/channel recognition are also estimated. The results are complementary to those for phonetic classification. Our results show how the phonetic information is locally spread and how the speaker/channel information is globally spread in time and frequency.

## 1  Introduction

Speech signals typically carry information about number of target sources such as linguistic message, speaker identity, and environment in which the speech was produced. In most realistic applications of speech technology, only one or a few information targets are important. For example, one may be interested in identifying the message in the signal regardless of the speaker or the environments in which the speech was produced, or the identification of the speaker is needed regardless of the words the targeted speaker is saying. Thus, not all components of the signal may be equally relevant for a decoding of the targeted information in the signal.

The speech research community has at its disposal rather large speech databases which are mainly used for training and testing automatic speech recognition (ASR) systems. There have been relatively few efforts to date to use such databases for deriving reusable knowledge about speech and speech communication processes which could be used for improvements of ASR technology. In this paper we apply information-theoretic approaches to study a large hand-labeled data set of fluent speech to learn about the information structure of the speech signal including the distribution of speech information in frequency and in time.

Based on the labeled data set, we analyze the relevancy of the features for phonetic

classifications and speaker/channel variability. The features in this data set are labeled with respect to underlying phonetic classes and files from which the features come from. The phoneme labels relate to the linguistic message in the signal, and the file labels carry the information about speakers and communication channels (each file contains speech of a single speaker transmitted through one telephone channel). Thus, phoneme and file labels are two target variables for statistical inference. The phoneme labels take 19 different values corresponding to 19 broad phoneme categories in the OGI Stories database [2]. The file labels take different values representing different speakers in the OGI Stories database.

The relevancy of a set of features is measured by the joint mutual information (JMI) between the features and a target variable. The phoneme target variable represents in our case the linguistic message. The file target variable represents both different speakers and different telephone channels. The joint mutual information between a target variable and the features quantifies the relevancy of the features for that target variable.

Mutual information measure the statistical dependence between random variables. Morris et al (1993) used mutual information to find the critical points of information for classifying French Vowel-Plosive-Vowel utterances. Bilmes(1998) showed recently that the information appears to be spread over relatively long temporal spans. While Bilmes used mutual information between two variables on non-labeled data to reveal the mutual dependencies between the components of the spectral energies in time and frequency, we focused on joint mutual information between the phoneme labels or file labels and one, two or three feature variables in the time-frequency plane[7, 6] and used this concept to gain insight into how information about phonemes and speaker/channel variability is distributed in the time-frequency plane.

## 2   Data Set and Preprocessing

The data set used in this paper is 3-hour phonetically labeled telephone speech, a subset of the English portion (Stories) of the OGI multi-lingual database [2] containing approximately 50 seconds of extemporaneous speech from each of 210 different speakers. The speech data is labeled by a variable $Y$ taking 19 values representing 19 most often occurring phoneme categories. The average phoneme duration is about 65 ms and the number of phoneme instances is 65421.

Acoustic features $X(f_k, t)$ for the experiments are derived from a short-time analysis of the speech signal with a 20 ms analysis window (Hamming) at the frame $t$ advanced in 10 ms steps. The logarithmic energy at a frequency $f_k$ is computed from the squared magnitude FFT using a critical-band spaced (log-like in the frequency variable) weighting function in a manner similar to that of the computation of Perceptual Linear Prediction coefficients [3]. In particular, the 5-th, 8-th and 12-th bands are centered around 0.5, 1 and 2 kHz respectively. Each feature $X(f_k, t)$ is labeled by a phoneme label $Y^p(t)$ and a file label $Y^f(t)$. We use mutual information to measure the relevancy of $X(f_k, t - d)$ across all frequencies $f_k$ and in a context window $-D \leq d \leq +D$ for the phoneme classification and the speaker/channel identification.

## 3   Estimation of MI and Bias Correction

In this paper, we only consider the mutual information (MI) between discrete random variables. The phoneme label and the file label are discrete random variables.

However, the feature variables are bounded continuous variables. To obtain the quantized features, we divide the maximum range of the observed features into cells of equal volume so that we can use histogram to estimate mutual information defined by

$$I(X;Y) = \sum_{x,y} p(x,y) \log_2 \frac{p(x,y)}{p(x)p(y)}.$$

If $X$ and $Y$ are jointly Gaussian, then $I(X;Y) = -\frac{1}{2}\ln(1-\rho^2)$ where $\rho$ is the correlation coefficient between $X$ and $Y$. However, for speech data the feature variables are generally non-Gaussian and target variables are categorical type variables. Correlations involving a categorical variable are meaningless.

The MI can also be written as

$$\begin{aligned} I(X;Y) &= H(X) + H(Y) - H(X,Y) \\ &= H(Y) - H(Y|X) = H(X) - H(X|Y) \end{aligned} \tag{1}$$

where $H(Y|X)$ is a conditional entropy defined by

$$H(Y|X) = -\sum_x p(x) \sum_y p(y|x) \log_2 p(y|x).$$

The two equations in (1) mean that the MI is the uncertainty reduction about $Y$ give $X$ or the uncertainty reduction about $X$ give $Y$.

Based on the histogram, $H(X)$ is estimated by

$$\hat{H}(X) = -\sum_i \frac{n_i}{n} \log_2 \frac{n_i}{n}$$

where $n_i$ is the number of data points in the i-th cell and $n$ is the data size. And $I(X;Y)$ is estimated by

$$\hat{I}(X;Y) = \hat{H}(X) + \hat{H}(Y) - \hat{H}(X,Y).$$

Miller(1954)[4] has shown that $\hat{H}(X)$ is an underestimate of $H(X)$ and $\hat{I}(X;Y)$ is an overestimate of $I(X;Y)$. The biases are

$$E[\hat{H}(X)] - H(X) = -\frac{r-1}{2\ln(2)n} + O(\frac{1}{n^2}) \tag{2}$$

$$E[\hat{I}(X;Y)] - I(X;Y) = \frac{(r-1)(c-1)}{2\ln(2)n} + O(\frac{1}{n^2}) \tag{3}$$

where $r$ and $c$ are the number of cells for $X$ and $Y$ respectively.

Interestingly, the first order terms in (2) and (3) do not depend on the probability distribution. After using these formulas to correct the estimates, the new estimates have the same variances as the old estimates but with reduced biases. However, these formulas break down when $r$ and $n$ are of the same order. Extending Miller's approach, we find a high order correction for the bias. Let $\{p_i\}$ be the probability distribution of $X$, then

$$\begin{aligned} E[\hat{H}(X)] - H(X) &= -\frac{r-1}{2\ln(2)n} + \frac{1}{6\ln(2)n^2}(S(\{p_i\}) - 3r + 2) \\ &\quad - \frac{1}{4n^3}(S(\{p_i\}) - 1) + O(\frac{1}{n^4}) \end{aligned} \tag{4}$$

where $S(\{p_i\}) = \sum_{i=1, p_i \neq 0}^r \frac{1}{p_i}$.

The last two terms in the bias (4) depend on the unknown probabilities $\{p_i\}$. In practice they are approximated by the relative frequency estimates.

Similarly, we can find the bias formulas of the high order terms $O(\frac{1}{n^2})$ and $O(\frac{1}{n^3})$ for the MI estimate.

When $X$ is evenly distributed, $p_i = 1/r$, so $S(\{p_i\}) = r^2$ and

$$E[\hat{H}(X)] - H(X) = -\frac{r-1}{2\ln(2)n} + \frac{1}{6\ln(2)n^2}(r^2 - 3r + 2) - \frac{1}{4n^3}(r^2 - 1) + O(\frac{1}{n^4}).$$

Theoretically $S(\{p_i\})$ has no upper bound when one of the probabilities is close to zero. However, in practice it is hard to collect a sample to estimate a very small probability. For this reason, we assume that $p_i$ is either zero or greater than $\varepsilon/r$ where $\varepsilon > 0$ is a small constant does not depend on $n$ or $r$. Under this assumption $S(\{p_i\}) \leq r^2/\varepsilon$ and the amplitude of the last term in (4) is less than $\frac{1}{4n^3}(r^2/\varepsilon - 1)$.

## 4   MI in Speech for Phonetic Classification

The three hour telephone speech in the OGI database gives us a sample size greater than 1 million, $n = 1050000$. To estimate the mutual information between three features and a target variable, we need to estimate the entropy $H(X_1, X_2, X_3, Y)$. Take $B = 20$ as the number of bins for each feature variable and $C = 19$ is the number of phoneme categories. Then the total number of cells is $r = B^3 * C$. After a constant adjustment, assuming $\varepsilon = 1$, the bias is

$$O(\frac{1}{n^2}) = \frac{1}{6\ln(2)n^2}(r^2 - 3r + 2) = 0.005(\text{bits}).$$

It is shown in Fig. 1(a) that $X(f_4, t)$ and $X(f_5, t)$ are most relevant features for phonetic classification. From Fig. 1(b), at 5 Bark the MI spread around the current frame is 200 ms.

Given one feature $X_1$, the information gain due to the second feature is the difference

$$I(X_1, X_2; Y) - I(X_1; Y) = I(X_2; Y|X_1)$$

where $I(X_2; Y|X_1)$ is called the information gain of $X_2$ given $X_1$. It is a conditional mutual information defined by

$$I(X_2; Y|X_1) = \sum_{x_1} p(x_1) \sum_{x_2, y} p(x_2, y|x_1) \log_2 \frac{p(x_2, y|x_1)}{p(x_2|x_1)p(y|x_1)}$$

It is shown in Fig. 1(c)-(d) that given $X(f_5, t)$ across different bands the maximum information gain is achieved by $X(f_9, t)$, and within 5 Bark band the maximum information gain is achieved by $X(f_5, t - 5)$. The mutual informations $I(X(f_4, t), X(f_k, t+d); Y)$ for $k = 1, \cdots, 15, k \neq 4$, and $d = \pm 1, \cdots, \pm 10$, the information gain from the second feature in the vicinity of the first one, are shown in Fig. 2. The asymmetric distribution of the MI around the neighborhood $(f_5, d = 0)$ indicates that the phonetic information is spread asymmetrically through time but localized in about 200 ms around the current frame.

Based on our data set, we have $\hat{H}(Y) = 3.96$ (bits). The JMI for three frequency features and three temporal features are shown in Fig. 1(e)-(f). Based on these estimates, the three frequency features give 28% reduction in uncertainty about $Y$ while the three temporal features give 19% reduction.

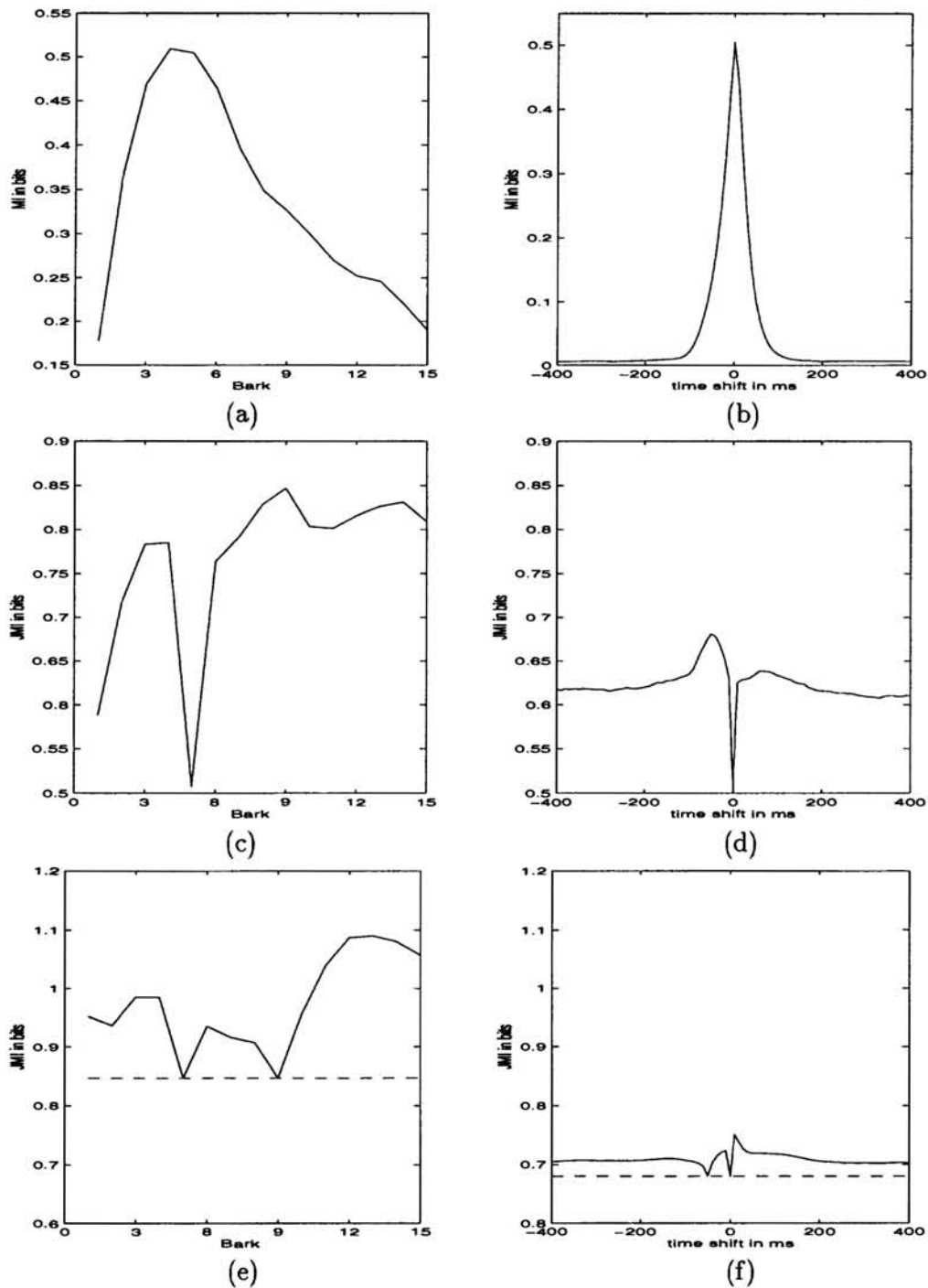

Figure 1: (a) MIs of individual features in different bands. (b) MIs of individual feature at 5 Bark with different 10ms-frame shifts. (c) JMIs of two features: at 5 Bark and in other bands. (d) JMIs of two features: current frame and shifted frames, both at 5 Bark. (e) JMIs of three features: at 5 Bark, 9 Bark and in other bands. The dashed line is the JMI level achieved by the two features $X(f_5,t)$ and $X(f_9,t)$. (f) JMIs of three features: current frame, 5th frame before current frame, and other shifted frames, all at 5 Bark. The dashed line is the JMI level achieved by $X(f_5,t)$ and $X(f_5,t-5)$.

The size of our data set is $n = 1050000$. Therefore, we can reliably estimate the joint

MI between three features and the phoneme label. However, to estimate the JMI for more than 3 features we have the problem of curse of dimensionality since for $k$ features, $r = B^k * C$ is exponential increasing. For example, when $k = 4, B = 20$, and $C = 19$, the second order bias is $O(1/n^2) = 2.02$ (bits) which is too high to be ignored. To extend our approach beyond the current three-feature level, we need either to enlarge our data set or to find an alternative to the histogram based MI estimation.

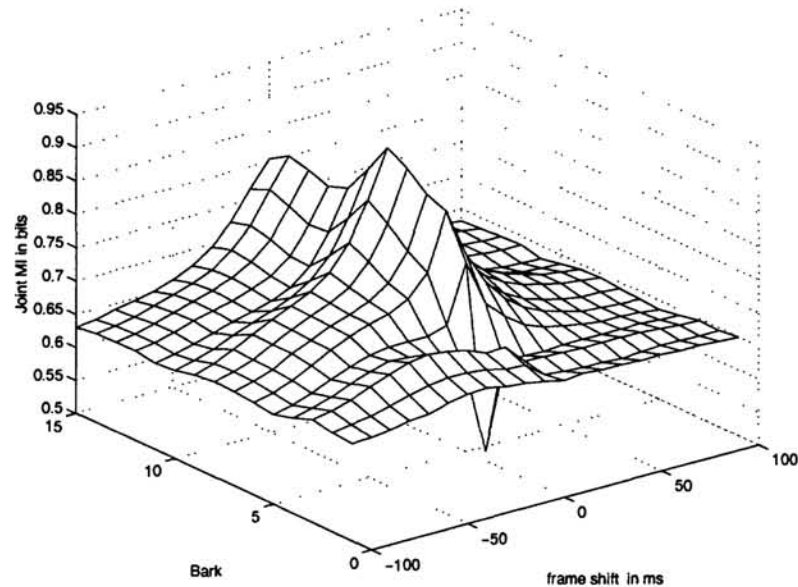

Figure 2: The 3-D plot of joint mutual information around $X(f_4, t)$. An asymmetric distribution is apparent especially around 4 Bark and 5 Bark.

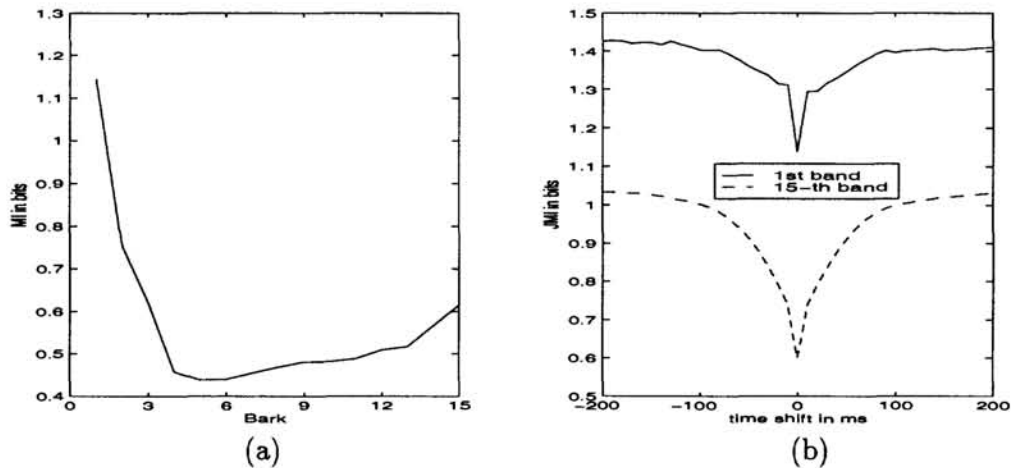

Figure 3: (a) The MI between one frequency feature and the file label. (b) The JMI between two features and the file identity labels.

## 5  MI in Speech for Speaker/Channel Recognition

The linguistic variability expressed by phoneme labels is not the only variability present in speech. We use the mutual information to evaluate relevance to other

sources of variabilities such as speaker/channel variability. Taking the file label as a target variable, we estimated the mutual information for one and two features.

It is shown in Fig. 3(a) that the most relevant features are in the very low frequency channels, which in our case of telephone speech carry only very little speech information. Fig. 3(b) shows that the second most relevant feature for speaker/channel recognition is at least 150 ms apart from the first most relevant feature. These results suggest that the information about the speaker and the communication channel is not localized in time. These results are complementary to the results for phonetic classification shown in Fig. 1(a) and (d).

## 6 CONCLUSIONS

Our results have shown that the information theoretic analysis of labeled speech data is feasible and useful for obtaining reusable knowledge about speech/channel variabilities. The joint mutual information of two features for phonetic classification is asymmetric around the current frame. We also estimated the joint mutual information between the phoneme labels and three feature variables. The uncertainty about the phonetic classification is reduced by adding more features. The maximum uncertainty reductions due to three frequency features and three temporal features are 28% and 19% respectively.

The mutual informations of one and two features for speaker/channel recognition are estimated. The results show that the most relevant features are in the very low frequency bands. At 1 Bark and 5 Bark, the second most relevant temporal feature for speaker/channel recognition is at least 150 ms apart from the first most relevant feature. These results suggest that the information about the speaker and the communication channel is not localized in time. These results are complementary to the results for phonetic classification for which the mutual information is generally localized with some time spread.

## References

[1] J. A. Bilmes. Maximum mutual information based reduction strategies for cross-correlation based joint distribution modeling. In *ICASSP98*, pages 469–472, April 1998.

[2] R. Cole, M. Fanty, M. Noel, and T. Lander. Telephone speech corpus development at CSLU. In *ICSLP*, pages 1815–1818, Yokohama, Sept. 1994.

[3] H. Hermansky. Perceptual linear predictive (PLP) analysis of speech. *J. Acoust. Soc. Am.*, 87(4):1738–1752, April 1990.

[4] G. A. Miller. Note on the bias of information estimates. In H. Quastler, editor, *Information Theory and Psychology*, pages 95–100. The Free Press, Illinois, 1954.

[5] Andrew Morris, Jean-Luc Schwartz, and Pierre Escudier. An information theoretical investigation into the distribution of phonetic information across the auditory spectogram. *Computer Speech & Language*, 7(2):121–136, April 1993.

[6] H. H. Yang, S. Van Vuuren, , S. Sharma, and H. Hermansky. Relevancy of time-frequency features for phonetic classification and speaker-channel recognition. *Accepted by Speech Communication*, 1999.

[7] H. H. Yang, S. Van Vuuren, and H. Hermansky. Relevancy of time-frequency features for phonetic classification measured by mutual information. In *ICASSP99*, pages I:225–228, Phoenix, March 1999.
